# Online Passive-Aggressive Algorithms

**Koby Crammer   Ofer Dekel   Shai Shalev-Shwartz   Yoram Singer**
School of Computer Science & Engineering
The Hebrew University, Jerusalem 91904, Israel
{kobics,oferd,shais,singer}@cs.huji.ac.il

## Abstract

We present a unified view for *online* classification, regression, and uni-class problems. This view leads to a single algorithmic framework for the three problems. We prove worst case loss bounds for various algorithms for both the realizable case and the non-realizable case. A conversion of our main online algorithm to the setting of batch learning is also discussed. The end result is new algorithms and accompanying loss bounds for the hinge-loss.

## 1   Introduction

In this paper we describe and analyze several learning tasks through the same algorithmic prism. Specifically, we discuss online classification, online regression, and online uniclass prediction. In all three settings we receive instances in a sequential manner. For concreteness we assume that these instances are vectors in $\mathbb{R}^n$ and denote the instance received on round $t$ by $\mathbf{x}_t$. In the classification problem our goal is to find a mapping from the instance space into the set of labels, $\{-1, +1\}$. In the regression problem the mapping is into $\mathbb{R}$. Our goal in the uniclass problem is to find a center-point in $\mathbb{R}^n$ with a small Euclidean distance to all of the instances.

We first describe the classification and regression problems. For classification and regression we restrict ourselves to mappings based on a weight vector $\mathbf{w} \in \mathbb{R}^n$, namely the mapping $f : \mathbb{R}^n \to \mathbb{R}$ takes the form $f(\mathbf{x}) = \mathbf{w} \cdot \mathbf{x}$. After receiving $\mathbf{x}_t$ we extend a prediction $\hat{y}_t$ using $f$. For regression the prediction is simply $\hat{y}_t = f(\mathbf{x}_t)$ while for classification $\hat{y}_t = \text{sign}(f(\mathbf{x}_t))$. After extending the prediction $\hat{y}_t$, we receive the true outcome $y_t$. We then suffer an *instantaneous* loss based on the discrepancy between $y_t$ and $f(\mathbf{x}_t)$. The goal of the online learning algorithm is to minimize the *cumulative* loss. The losses we discuss in this paper depend on a pre-defined insensitivity parameter $\epsilon$ and are denoted $\ell_\epsilon(\mathbf{w}; (\mathbf{x}, y))$. For regression the $\epsilon$-insensitive loss is,

$$\ell_\epsilon(\mathbf{w}; (\mathbf{x}, y)) = \begin{cases} 0 & |y - \mathbf{w} \cdot \mathbf{x}| \le \epsilon \\ |y - \mathbf{w} \cdot \mathbf{x}| - \epsilon & \text{otherwise} \end{cases} , \tag{1}$$

while for classification the $\epsilon$-insensitive loss is defined to be,

$$\ell_\epsilon(\mathbf{w}; (\mathbf{x}, y)) = \begin{cases} 0 & y(\mathbf{w} \cdot \mathbf{x}) \ge \epsilon \\ \epsilon - y(\mathbf{w} \cdot \mathbf{x}) & \text{otherwise} \end{cases} . \tag{2}$$

As in other online algorithms the weight vector $\mathbf{w}$ is updated after receiving the feedback $y_t$. Therefore, we denote by $\mathbf{w}_t$ the vector used for prediction on round $t$. We leave the details on the form this update takes to later sections.

| Problem | Example $(\mathbf{z}_t)$ | Discrepancy $(\delta)$ | Update Direction $(\mathbf{v}_t)$ |
|---|---|---|---|
| Classification | $(\mathbf{x}_t, y_t) \in \mathbb{R}^n \times \{-1, +1\}$ | $-y_t(\mathbf{w}_t \cdot \mathbf{x}_t)$ | $y_t \mathbf{x}_t$ |
| Regression | $(\mathbf{x}_t, y_t) \in \mathbb{R}^n \times \mathbb{R}$ | $\|y_t - \mathbf{w}_t \cdot \mathbf{x}_t\|$ | $\text{sign}(y_t - \mathbf{w}_t \cdot \mathbf{x}_t)\, \mathbf{x}_t$ |
| Uniclass | $(\mathbf{x}_t, y_t) \in \mathbb{R}^n \times \{1\}$ | $\|\mathbf{x}_t - \mathbf{w}_t\|$ | $\frac{\mathbf{x}_t - \mathbf{w}_t}{\|\mathbf{x}_t - \mathbf{w}_t\|}$ |

Table 1: Summary of the settings and parameters employed by the additive PA algorithm for classification, regression, and uniclass.

The setting for uniclass is slightly different as we only observe a sequence of instances. The goal of the uniclass algorithm is to find a center-point $\mathbf{w}$ such that all instances $\mathbf{x}_t$ fall within a radius of $\epsilon$ from $\mathbf{w}$. Since we employ the framework of online learning the vector $\mathbf{w}$ is constructed incrementally. The vector $\mathbf{w}_t$ therefore plays the role of the *instantaneous* center and is adapted after observing each instance $\mathbf{x}_t$. If an example $\mathbf{x}_t$ falls within a Euclidean distance $\epsilon$ from $\mathbf{w}_t$ then we suffer no loss. Otherwise, the loss is the distance between $\mathbf{x}_t$ and a ball of radius $\epsilon$ centered at $\mathbf{w}_t$. Formally, the uniclass loss is,

$$\ell_\epsilon(\mathbf{w}_t; \mathbf{x}_t) = \begin{cases} 0 & \|\mathbf{x}_t - \mathbf{w}_t\| \leq \epsilon \\ \|\mathbf{x}_t - \mathbf{w}_t\| - \epsilon & \text{otherwise} \end{cases}. \tag{3}$$

In the next sections we give additive and multiplicative online algorithms for the above learning problems and prove respective online loss bounds. A common thread of our approach is a unified view of all three tasks which leads to a single algorithmic framework with a common analysis.

**Related work:** Our work builds on numerous techniques from online learning. The updates we derive are based on an optimization problem directly related to the one employed by Support Vector Machines [15]. Li and Long [14] were among the first to suggest the idea of converting a batch optimization problem into an online task. Our work borrows ideas from the work of Warmuth and colleagues [11]. In particular, Gentile and Warmuth [6] generalized and adapted techniques from [11] to the hinge loss which is closely related to the losses defined in Eqs. (1)-(3). Kivinen et al. [10] discussed a general framework for gradient-based online learning where some of their bounds bare similarities to the bounds presented in this paper. Our work also generalizes and greatly improves online loss bounds for classification given in [3]. Herbster [8] suggested an algorithm for classification and regression that is equivalent to one of the algorithms given in this paper, however, the loss-bound derived by Herbster is somewhat weaker. Finally, we would like to note that similar algorithms have been devised in the convex optimization community (cf. [1, 2]). The main difference between these algorithms and the online algorithms presented in this paper lies in the analysis: while we derive worst case, finite horizon loss bounds, the optimization community is mostly concerned with asymptotic convergence properties.

## 2 A Unified Loss

The three problems described in the previous section share common algebraic properties which we explore in this section. The end result is a common algorithmic framework that is applicable to all three problems and an accompanying analysis (Sec. 3). Let $\mathbf{z}_t = (\mathbf{x}_t, y_t)$ denote the instance-target pair received on round $t$ where in the case of uniclass we set $y_t = 1$ as a placeholder. For a given example $\mathbf{z}_t$, let $\delta(\mathbf{w}; \mathbf{z}_t)$ denote the discrepancy of $\mathbf{w}$ on $\mathbf{z}_t$: for classification we set the discrepancy to be $-y_t(\mathbf{w}_t \cdot \mathbf{x}_t)$ (the negative of the margin), for regression it is $|y_t - \mathbf{w}_t \cdot \mathbf{x}_t|$, and for uniclass $\|\mathbf{x}_t - \mathbf{w}_t\|$. Fixing $\mathbf{z}_t$, we also

view $\delta(\mathbf{w}; \mathbf{z}_t)$ as a *convex* function of $\mathbf{w}$. Let $[a]_+$ be the function that equals $a$ whenever $a > 0$ and otherwise equals zero. Using the discrepancies defined above, the three different losses given in Eqs. (1)-(3) can all be written as $\ell_\epsilon(\mathbf{w}; \mathbf{z}) = [\delta(\mathbf{w}; \mathbf{z}) - \epsilon]_+$, where for classification we set $\epsilon \leftarrow -\epsilon$ since the discrepancy is defined as the negative of the margin. While this construction might seem a bit odd for classification, it is very useful in unifying the three problems. To conclude, the loss in all three problems can be derived by applying the same hinge loss to different (problem dependent) discrepancies.

## 3  An Additive Algorithm for the Realizable Case

Equipped with the simple unified notion of loss we describe in this section a single online algorithm that is applicable to all three problems. The algorithm and the analysis we present in this section assume that there exist a weight vector $\mathbf{w}^\star$ and an insensitivity parameter $\epsilon^\star$ for which the data is perfectly realizable. Namely, we assume that $\ell_{\epsilon^\star}(\mathbf{w}^\star; \mathbf{z}_t) = 0$ for all $t$ which implies that,

$$y_t(\mathbf{w}^\star \cdot \mathbf{x}_t) \geq |\epsilon^\star| \ \text{(Class.)} \quad |y_t - \mathbf{w}^\star \cdot \mathbf{x}_t| \leq \epsilon^\star \ \text{(Reg.)} \quad \|\mathbf{x}_t - \mathbf{w}^\star\| \leq \epsilon^\star \ \text{(Unic.)} \ . \ (4)$$

A modification of the algorithm for the unrealizable case is given in Sec. 5.

The general method we use for deriving our on-line update rule is to define the new weight vector $\mathbf{w}_{t+1}$ as the solution to the following projection problem

$$\mathbf{w}_{t+1} = \underset{\mathbf{w}}{\operatorname{argmin}} \quad \frac{1}{2}\|\mathbf{w} - \mathbf{w}_t\|^2 \quad \text{s.t.} \quad \ell_\epsilon(\mathbf{w}; \mathbf{z}_t) = 0 \ , \tag{5}$$

namely, $\mathbf{w}_{t+1}$ is set to be the projection of $\mathbf{w}_t$ onto the set of all weight vectors that attain a loss of zero. We denote this set by $C$. For the case of classification, $C$ is a half space, $C = \{\mathbf{w} : -y_t\mathbf{w} \cdot \mathbf{x}_t \leq \epsilon\}$. For regression $C$ is an $\epsilon$-hyper-slab, $C = \{\mathbf{w} : |\mathbf{w} \cdot \mathbf{x}_t - y_t| \leq \epsilon\}$ and for uniclass it is a ball of radius $\epsilon$ centered at $\mathbf{x}_t$, $C = \{\mathbf{w} : \|\mathbf{w} - \mathbf{x}_t\| \leq \epsilon\}$. In Fig. 2 we illustrate the projection for the three cases. This optimization problem attempts to keep $\mathbf{w}_{t+1}$ as close to $\mathbf{w}_t$ as possible, while forcing $\mathbf{w}_{t+1}$ to achieve a zero loss on the most recent example. The resulting algorithm is *passive* whenever the loss is zero, that is, $\mathbf{w}_{t+1} = \mathbf{w}_t$ whenever $\ell_\epsilon(\mathbf{w}_t; \mathbf{z}_t) = 0$. In contrast, on rounds for which $\ell_\epsilon(\mathbf{w}_t; \mathbf{z}_t) > 0$ we *aggressively* force $\mathbf{w}_{t+1}$ to satisfy the constraint $\ell_\epsilon(\mathbf{w}_{t+1}; \mathbf{z}_t) = 0$.

Therefore we name the algorithm passive-aggressive or PA for short. In the following we show that for the three problems described above the solution to the optimization problem in Eq. (5) yields the following update rule,

$$\mathbf{w}_{t+1} = \mathbf{w}_t + \tau_t \mathbf{v}_t \ , \tag{6}$$

where $\mathbf{v}_t$ is minus the gradient of the discrepancy and $\tau_t = \ell_\epsilon(\mathbf{w}_t; \mathbf{z}_t)/\|\mathbf{v}_t\|^2$. (Note that although the discrepancy might not be differentiable everywhere, its gradient exists whenever the loss is greater than zero). To see that the

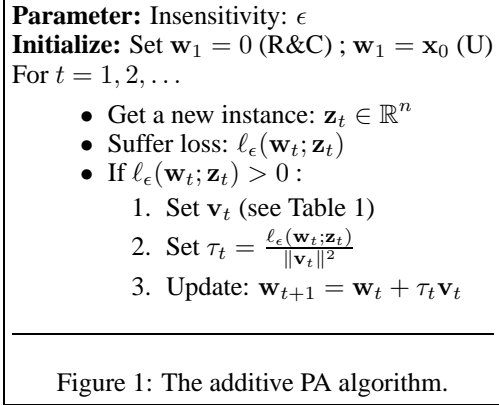

**Parameter:** Insensitivity: $\epsilon$
**Initialize:** Set $\mathbf{w}_1 = 0$ (R&C) ; $\mathbf{w}_1 = \mathbf{x}_0$ (U)
For $t = 1, 2, \ldots$

- Get a new instance: $\mathbf{z}_t \in \mathbb{R}^n$
- Suffer loss: $\ell_\epsilon(\mathbf{w}_t; \mathbf{z}_t)$
- If $\ell_\epsilon(\mathbf{w}_t; \mathbf{z}_t) > 0$:
  1. Set $\mathbf{v}_t$ (see Table 1)
  2. Set $\tau_t = \frac{\ell_\epsilon(\mathbf{w}_t; \mathbf{z}_t)}{\|\mathbf{v}_t\|^2}$
  3. Update: $\mathbf{w}_{t+1} = \mathbf{w}_t + \tau_t \mathbf{v}_t$

Figure 1: The additive PA algorithm.

update from Eq. (6) is the solution to the problem defined by Eq. (5), first note that the equality constraint $\ell_\epsilon(\mathbf{w}; \mathbf{z}_t) = 0$ is equivalent to the inequality constraint $\delta(\mathbf{w}; \mathbf{z}_t) \leq \epsilon$. The Lagrangian of the optimization problem is

$$\mathcal{L}(\mathbf{w}, \tau) = \frac{1}{2}\|\mathbf{w} - \mathbf{w}_t\|^2 + \tau\left(\delta(\mathbf{w}; \mathbf{z}_t) - \epsilon\right) \ , \tag{7}$$

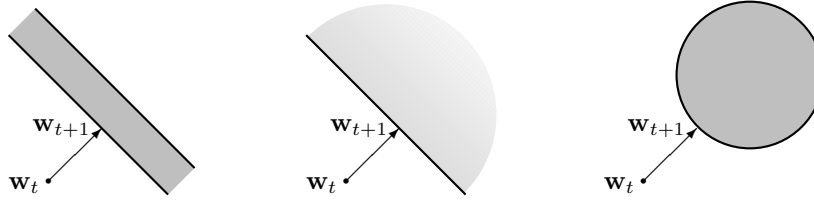

Figure 2: An illustration of the update: $\mathbf{w}_{t+1}$ is found by projecting the current vector $\mathbf{w}_t$ onto the set of vectors attaining a zero loss on $\mathbf{z}_t$. This set is a stripe in the case of regression, a half-space for classification, and a ball for uniclass.

where $\tau \geq 0$ is a Lagrange multiplier. To find a saddle point of $\mathcal{L}$ we first differentiate $\mathcal{L}$ with respect to $\mathbf{w}$ and use the fact that $\mathbf{v}_t$ is minus the gradient of the discrepancy to get,

$$\nabla_{\mathbf{w}}(\mathcal{L}) \;=\; \mathbf{w} - \mathbf{w}_t + \tau\nabla_{\mathbf{w}}\delta \;=\; 0 \quad \Rightarrow \quad \mathbf{w} \;=\; \mathbf{w}_t + \tau\mathbf{v}_t \;.$$

To find the value of $\tau$ we use the KKT conditions. Hence, whenever $\tau$ is positive (as in the case of non-zero loss), the inequality constraint, $\delta(\mathbf{w}; \mathbf{z}_t) \leq \epsilon$, becomes an equality. Simple algebraic manipulations yield that the value $\tau$ for which $\delta(\mathbf{w}; \mathbf{z}_t) = \epsilon$ for all three problems is equal to, $\tau_t = \ell_\epsilon(\mathbf{w}; \mathbf{z}_t)/\|\mathbf{v}_t\|^2$. A summary of the discrepancy functions and their respective updates is given in Table 1. The pseudo-code of the additive algorithm for all three settings is given in Fig. 1.

We now discuss the initialization of $\mathbf{w}_1$. For classification and regression a reasonable choice for $\mathbf{w}_1$ is the zero vector. However, in the case of uniclass initializing $\mathbf{w}_1$ to be the zero vector might incur large losses if, for instance, all the instances are located far away from the origin. A more sensible choice for uniclass is to initialize $\mathbf{w}_1$ to be one of the examples. For simplicity of the description we assume that we are provided with an example $\mathbf{x}_0$ prior to the run of the algorithm and initialize $\mathbf{w}_1 = \mathbf{x}_0$.

To conclude this section we note that for all three cases the weight vector $\mathbf{w}_t$ is a linear combination of the instances. This representation enables us to employ kernels [15].

## 4  Analysis

The following theorem provides a unified loss bound for all three settings. After proving the theorem we discuss a few of its implications.

**Theorem 1** *Let* $\mathbf{z}_1, \mathbf{z}_2, \ldots, \mathbf{z}_t, \ldots$ *be a sequence of examples for one of the problems described in Table 1. Assume that there exist* $\mathbf{w}^\star$ *and* $\epsilon^\star$ *such that* $\ell_{\epsilon^\star}(\mathbf{w}^\star; \mathbf{z}_t) = 0$ *for all* $t$. *Then if the additive PA algorithm is run with* $\epsilon \geq \epsilon^\star$, *the following bound holds for any* $T \geq 1$

$$\sum_{t=1}^{T} \left(\ell_\epsilon(\mathbf{w}_t; \mathbf{z}_t)\right)^2 \;\;+\;\; 2(\epsilon - \epsilon^\star)\sum_{t=1}^{T}\ell_\epsilon(\mathbf{w}_t; \mathbf{z}_t) \;\;\leq\;\; B\,\|\mathbf{w}^\star - \mathbf{w}_1\|^2 \;, \qquad (8)$$

*where for classification and regression* $B$ *is a bound on the squared norm of the instances* $(\forall t : B \geq \|\mathbf{x}_t\|_2^2)$ *and* $B = 1$ *for uniclass.*

**Proof:** Define $\Delta_t = \|\mathbf{w}_t - \mathbf{w}^\star\|^2 - \|\mathbf{w}_{t+1} - \mathbf{w}^\star\|^2$. We prove the theorem by bounding $\sum_{t=1}^{T} \Delta_t$ from above and below. First note that $\sum_{t=1}^{T} \Delta_t$ is a telescopic sum and therefore

$$\sum_{t=1}^{T} \Delta_t \;=\; \|\mathbf{w}_1 - \mathbf{w}^\star\|^2 - \|\mathbf{w}_{T+1} - \mathbf{w}^\star\|^2 \;\leq\; \|\mathbf{w}_1 - \mathbf{w}^\star\|^2 \;. \tag{9}$$

This provides an upper bound on $\sum_t \Delta_t$. In the following we prove the lower bound

$$\Delta_t \geq \frac{\ell_\epsilon(\mathbf{w}_t; \mathbf{z}_t)}{B} \left(\ell_\epsilon(\mathbf{w}_t; \mathbf{z}_t) + 2(\epsilon - \epsilon^\star)\right) \;. \tag{10}$$

First note that we do not modify $\mathbf{w}_t$ if $\ell_\epsilon(\mathbf{w}_t; \mathbf{z}_t) = 0$. Therefore, this inequality trivially holds when $\ell_\epsilon(\mathbf{w}_t; \mathbf{z}_t) = 0$ and thus we can restrict ourselves to rounds on which the discrepancy is larger than $\epsilon$, which implies that $\ell_\epsilon(\mathbf{w}_t; \mathbf{z}_t) = \delta(\mathbf{w}_t; \mathbf{z}_t) - \epsilon$. Let $t$ be such a round then by rewriting $\mathbf{w}_{t+1}$ as $\mathbf{w}_t + \tau_t \mathbf{v}_t$ we get,

$$\begin{aligned}
\Delta_t &= \|\mathbf{w}_t - \mathbf{w}^\star\|^2 - \|\mathbf{w}_{t+1} - \mathbf{w}^\star\|^2 \;=\; \|\mathbf{w}_t - \mathbf{w}^\star\|^2 - \|\mathbf{w}_t + \tau_t \mathbf{v}_t - \mathbf{w}^\star\|^2 \\
&= \|\mathbf{w}_t - \mathbf{w}^\star\|^2 - \left(\tau_t^2 \|\mathbf{v}_t\|^2 + 2\tau_t(\mathbf{v}_t \cdot (\mathbf{w}_t - \mathbf{w}^\star)) + \|\mathbf{w}_t - \mathbf{w}^\star\|^2\right) \\
&= -\tau_t^2 \|\mathbf{v}_t\|^2 + 2\tau_t \mathbf{v}_t \cdot (\mathbf{w}^\star - \mathbf{w}_t) \;.
\end{aligned} \tag{11}$$

Using the fact that $-\mathbf{v}_t$ is the gradient of the convex function $\delta(\mathbf{w}; \mathbf{z}_t)$ at $\mathbf{w}_t$ we have,

$$\delta(\mathbf{w}^\star; \mathbf{z}_t) - \delta(\mathbf{w}_t; \mathbf{z}_t) \geq (-\mathbf{v}_t) \cdot (\mathbf{w}^\star - \mathbf{w}_t) \;. \tag{12}$$

Adding and subtracting $\epsilon$ from the left-hand side of Eq. (12) and rearranging we get,

$$\mathbf{v}_t \cdot (\mathbf{w}^\star - \mathbf{w}_t) \;\geq\; \delta(\mathbf{w}_t; \mathbf{z}_t) - \epsilon + \epsilon - \delta(\mathbf{w}^\star; \mathbf{z}_t) \;. \tag{13}$$

Recall that $\delta(\mathbf{w}_t; \mathbf{z}_t) - \epsilon = \ell_\epsilon(\mathbf{w}_t; \mathbf{z}_t)$ and that $\epsilon^\star \geq \delta(\mathbf{w}^\star; \mathbf{z}_t)$. Therefore,

$$(\delta(\mathbf{w}_t; \mathbf{z}_t) - \epsilon) + (\epsilon - \delta(\mathbf{w}^\star; \mathbf{z}_t)) \geq \ell_\epsilon(\mathbf{w}_t; \mathbf{z}_t) + (\epsilon - \epsilon^\star) \;. \tag{14}$$

Combining Eq. (11) with Eqs. (13-14) we get

$$\begin{aligned}
\Delta_t &\geq -\tau_t^2 \|\mathbf{v}_t\|^2 + 2\tau_t \left(\ell_\epsilon(\mathbf{w}_t; \mathbf{z}_t) + (\epsilon - \epsilon^\star)\right) \\
&= \tau_t \left(-\tau_t \|\mathbf{v}_t\|^2 + 2\ell_\epsilon(\mathbf{w}_t; \mathbf{z}_t) + 2(\epsilon - \epsilon^\star)\right) \;.
\end{aligned} \tag{15}$$

Plugging $\tau_t = \ell_\epsilon(\mathbf{w}_t; \mathbf{z}_t)/\|\mathbf{v}_t\|^2$ into Eq. (15) we get

$$\Delta_t \geq \frac{\ell_\epsilon(\mathbf{w}_t; \mathbf{z}_t)}{\|\mathbf{v}_t\|^2} \left(\ell_\epsilon(\mathbf{w}_t; \mathbf{z}_t) + 2(\epsilon - \epsilon^\star)\right) \;.$$

For uniclass $\|\mathbf{v}_t\|^2$ is always equal to 1 by construction and for classification and regression we have $\|\mathbf{v}_t\|^2 = \|\mathbf{x}_t\|^2 \leq B$ which gives,

$$\Delta_t \geq \frac{\ell_\epsilon(\mathbf{w}_t; \mathbf{z}_t)}{B} \left(\ell_\epsilon(\mathbf{w}_t; \mathbf{z}_t) + 2(\epsilon - \epsilon^\star)\right) \;.$$

Comparing the above lower bound with the upper bound in Eq. (9) we get

$$\sum_{t=1}^{T} \left(\ell_\epsilon(\mathbf{w}_t; \mathbf{z}_t)\right)^2 + \sum_{t=1}^{T} 2(\epsilon - \epsilon^\star)\ell_\epsilon(\mathbf{w}_t; \mathbf{z}_t) \;\leq\; B\|\mathbf{w}^\star - \mathbf{w}_1\|^2 \;.$$

This concludes the proof. $\blacksquare$

Let us now discuss the implications of Thm. 1. We first focus on the classification case. Due to the realizability assumption, there exist $\mathbf{w}^\star$ and $\epsilon^\star$ such that for all $t$, $\ell_{\epsilon^\star}(\mathbf{w}^\star; \mathbf{z}_t) = 0$ which implies that $y_t(\mathbf{w}^\star \cdot \mathbf{x}_t) \geq -\epsilon^\star$. Dividing $\mathbf{w}^\star$ by its norm we can rewrite the latter as $y_t(\hat{\mathbf{w}}^\star \cdot \mathbf{x}_t) \geq \hat{\epsilon}^\star$ where $\hat{\mathbf{w}}^\star = \mathbf{w}^\star/\|\mathbf{w}^\star\|$ and $\hat{\epsilon}^\star = |\epsilon^\star|/\|\mathbf{w}^\star\|$. The parameter $\hat{\epsilon}^\star$ is often

referred to as the *margin* of a unit-norm separating hyperplane. Now, setting $\epsilon = -1$ we get that $\ell_\epsilon(\mathbf{w}; \mathbf{z}) = [1 - y(\mathbf{w} \cdot \mathbf{x})]_+$ – the hinge loss for classification. We now use Thm. 1 to obtain two loss bounds for the hinge loss in a classification setting. First, note that by also setting $\mathbf{w}^\star = \hat{\mathbf{w}}^\star / \hat{\epsilon}^\star$ and thus $\epsilon^\star = -1$ we get that the second term on the left hand side of Eq. (8) vanishes as $\epsilon^\star = \epsilon = -1$ and thus,

$$\sum_{t=1}^T \left([1 - y_t(\mathbf{w}_t \cdot \mathbf{x}_t)]_+\right)^2 \ \leq \ B \left\|\mathbf{w}^\star\right\|^2 \ = \ \frac{B}{(\hat{\epsilon}^\star)^2} \ . \tag{17}$$

We thus have obtained a bound on the squared hinge loss. The same bound was also derived by Herbster [8]. We can immediately use this bound to derive a mistake bound for the PA algorithm. Note that the algorithm makes a prediction mistake iff $y_t(\mathbf{w}_t \cdot \mathbf{x}_t) \leq 0$. In this case, $[1 - y_t(\mathbf{w}_t \cdot \mathbf{x}_t)]_+ \geq 1$ and therefore the number of prediction mistakes is bounded by $B/(\hat{\epsilon}^\star)^2$. This bound is common to online algorithms for classification such as ROMMA [14].

We can also manipulate the result of Thm. 1 to obtain a direct bound on the hinge loss. Using again $\epsilon = -1$ and omitting the first term in the left hand side of Eq. (8) we get,

$$2(-1 - \epsilon^\star) \sum_{t=1}^T [1 - y_t(\mathbf{w}_t \cdot \mathbf{x}_t)]_+ \ \leq \ B\|\mathbf{w}^\star\|^2 \ .$$

By setting $\mathbf{w}^\star = 2\hat{\mathbf{w}}^\star / \hat{\epsilon}^\star$, which implies that $\epsilon^\star = -2$, we can further simplify the above to get a bound on the cumulative hinge loss,

$$\sum_{t=1}^T [1 - y_t(\mathbf{w}_t \cdot \mathbf{x}_t)]_+ \ \leq \ 2\frac{B}{(\hat{\epsilon}^\star)^2} \ .$$

To conclude this section, we would like to point out that the PA online algorithm can also be used as a building block for a batch algorithm. Concretely, let $S = \{\mathbf{z}_1, \ldots, \mathbf{z}_m\}$ be a fixed training set and let $\beta \in \mathbb{R}$ be a small positive number. We start with an initial weight vector $\mathbf{w}_1$ and then invoke the PA algorithm as follows. We choose an example $\mathbf{z} \in S$ such that $\ell_\epsilon(\mathbf{w}_1; \mathbf{z})^2 > \beta$ and present $\mathbf{z}$ to the PA algorithm. We repeat this process and obtain $\mathbf{w}_2, \mathbf{w}_3, \ldots$ until the $T$'th iteration on which for all $\mathbf{z} \in S$, $\ell_\epsilon(\mathbf{w}_T; \mathbf{z})^2 \leq \beta$. The output of the batch algorithm is $\mathbf{w}_T$. Due to the bound of Thm. 1, $T$ is at most $\lceil B\|\mathbf{w}^\star - \mathbf{w}_1\|^2/\beta \rceil$ and by construction the loss of $\mathbf{w}_T$ on any $\mathbf{z} \in S$ is at most $\sqrt{\beta}$. Moreover, in the following lemma we show that the norm of $\mathbf{w}_T$ cannot be too large. Since $\mathbf{w}_T$ achieves a small empirical loss and its norm is small, it can be shown using classical techniques (cf. [15]) that the loss of $\mathbf{w}_T$ on unseen data is small as well.

**Lemma 2** *Under the same conditions of Thm. 1, the following bound holds for any $T \geq 1$*
$$\|\mathbf{w}_T - \mathbf{w}_1\| \leq 2 \|\mathbf{w}^\star - \mathbf{w}_1\| \ .$$

**Proof:** First note that the inequality trivially holds for $T = 1$ and thus we focus on the case $T > 1$. We use the definition of $\Delta_t$ from the proof of Thm. 1. Eq. (10) implies that $\Delta_t$ is non-negative for all $t$. Therefore, we get from Eq. (9) that

$$0 \ \leq \ \sum_{t=1}^{T-1} \Delta_t \ = \ \|\mathbf{w}_1 - \mathbf{w}^\star\|^2 - \|\mathbf{w}_T - \mathbf{w}^\star\|^2 \ . \tag{18}$$

Rearranging the terms in Eq. (18) we get that $\|\mathbf{w}_T - \mathbf{w}^\star\| \leq \|\mathbf{w}^\star - \mathbf{w}_1\|$. Finally, we use the triangle inequality to get the bound,

$$\begin{aligned} \|\mathbf{w}_T - \mathbf{w}_1\| \ &= \ \|(\mathbf{w}_T - \mathbf{w}^\star) + (\mathbf{w}^\star - \mathbf{w}_1)\| \\ &\leq \ \|\mathbf{w}_T - \mathbf{w}^\star\| + \|\mathbf{w}^\star - \mathbf{w}_1\| \ \leq \ 2 \|\mathbf{w}^\star - \mathbf{w}_1\| \ . \end{aligned}$$

This concludes the proof. ∎

# 5 A Modification for the Unrealizable Case

We now briefly describe an algorithm for the unrealizable case. This algorithm applies only to regression and classification problems. The case of uniclass is more involved and will be discussed in detail elsewhere. The algorithm employs two parameters. The first is the insensitivity parameter $\epsilon$ which defines the loss function as in the realizable case. However, in this case we do not assume that there exists $\mathbf{w}^\star$ that achieves zero loss over the sequence. We instead measure the loss of the online algorithm *relative* to the loss of any vector $\mathbf{w}^\star$. The second parameter, $\gamma > 0$, is a relaxation parameter. Before describing the effect of this parameter we define the update step for the unrealizable case. As in the realizable case, the algorithm is conservative. That is, if the loss on example $\mathbf{z}_t$ is zero then $\mathbf{w}_{t+1} = \mathbf{w}_t$. In case the loss is positive the update rule is $\mathbf{w}_{t+1} = \mathbf{w}_t + \tau_t \mathbf{v}_t$ where $\mathbf{v}_t$ is the same as in the realizable case. However, the scaling factor $\tau_t$ is modified and is set to,

$$\tau_t = \frac{\ell_\epsilon(\mathbf{w}_t; \mathbf{z}_t)}{\|\mathbf{v}_t\|^2 + \gamma} \ .$$

The following theorem provides a loss bound for the online algorithm *relative* to the loss of any fixed weight vector $\mathbf{w}^\star$.

**Theorem 3** *Let $z_1 = (\mathbf{x}_1, y_1), \mathbf{z}_2 = (\mathbf{x}_2, y_2), \ldots, \mathbf{z}_t = (\mathbf{x}_t, y_t), \ldots$ be a sequence of classification or regression examples. Let $\mathbf{w}^\star$ be any vector in $\mathbb{R}^n$. Then if the PA algorithm for the unrealizable case is run with $\epsilon$, and with $\gamma > 0$, the following bound holds for any $T \geq 1$ and a constant $B$ satisfying $B \geq \|\mathbf{x}_t\|^2$,*

$$\sum_{t=1}^{T} \left(\ell_\epsilon(\mathbf{w}_t; \mathbf{z}_t)\right)^2 \ \leq \ (\gamma + B)\|\mathbf{w}^\star - \mathbf{w}_1\|^2 \ + \ \left(1 + \frac{B}{\gamma}\right) \sum_{t=1}^{T} \left(\ell_\epsilon(\mathbf{w}^\star; \mathbf{z}_t)\right)^2 \ . \quad (19)$$

The proof of the theorem is based on a reduction to the realizable case (cf. [4, 13, 14]) and is omitted due to the lack of space.

# 6 Extensions

There are numerous potential extensions to our approach. For instance, if all the components of the instances are non-negative we can derive a multiplicative version of the PA algorithm. The multiplicative PA algorithm maintains a weight vector $\mathbf{w}_t \in \mathbb{P}^n$ where $\mathbb{P}^n = \{\mathbf{x} \ : \ \mathbf{x} \in \mathbb{R}^n_+, \ \sum_{j=1}^{n} \mathbf{x}_j = 1\}$. The multiplicative update of $\mathbf{w}_t$ is,

$$w_{t+1,j} \ = \ (1/Z_t)\, w_{t,j}\, e^{\tau_t v_{t,j}} \ ,$$

where $\mathbf{v}_t$ is the same as the one used in the additive algorithm (Table 1), $\tau_t$ now becomes $4\ell_\epsilon(\mathbf{w}_t; \mathbf{z}_t)/\|\mathbf{v}_t\|^2_\infty$ for regression and classification and $\ell_\epsilon(\mathbf{w}_t; \mathbf{z}_t)/(8\|\mathbf{v}_t\|^2_\infty)$ for uniclass and $Z_t = \sum_{j=1}^{n} w_{t,j} e^{\tau_t v_{t,j}}$ is a normalization factor. For the multiplicative PA we can prove the following loss bound.

**Theorem 4** *Let $\mathbf{z}_1, \mathbf{z}_2, \ldots, \mathbf{z}_t = (\mathbf{x}_t, y_t), \ldots$ be a sequence of examples such that $x_{t,j} \geq 0$ for all t. Let $D_{RE}(\mathbf{w}\|\mathbf{w}') = \sum_j w_j \log(w_j/w'_j)$ denote the relative entropy between $\mathbf{w}$ and $\mathbf{w}'$. Assume that there exist $\mathbf{w}^\star$ and $\epsilon^\star$ such that $\ell_{\epsilon^\star}(\mathbf{w}^\star; \mathbf{z}_t) = 0$ for all t. Then when the multiplicative version of the PA algorithm is run with $\epsilon > \epsilon^\star$, the following bound holds for any $T \geq 1$,*

$$\sum_{t=1}^{T} \left(\ell_\epsilon(\mathbf{w}_t; \mathbf{z}_t)\right)^2 \ + \ 2(\epsilon - \epsilon^\star) \sum_{t=1}^{T} \ell_\epsilon(\mathbf{w}_t; \mathbf{z}_t) \ \leq \ \frac{1}{2} B\, D_{RE}(\mathbf{w}^\star\|\mathbf{w}_1) \ ,$$

*where for classification and regression $B$ is a bound on the square of the infinity norm of the instances $(\forall t : B \geq \|\mathbf{x}_t\|^2_\infty)$ and $B = 16$ for uniclass.*

The proof of the theorem is rather technical and uses the proof technique of Thm. 1 in conjunction with inequalities on the logarithm of $Z_t$ (see for instance [7, 11, 9]).

An interesting question is whether the unified view of classification, regression, and uniclass can be exported and used with other algorithms for classification such as ROMMA [14] and ALMA [5]. Another, rather general direction for possible extension surfaces when replacing the Euclidean distance between $\mathbf{w}_{t+1}$ and $\mathbf{w}_t$ with other distances and divergences such as the Bregman divergence. The resulting optimization problem may be solved via Bregman projections. In this case it might be possible to derive general loss bounds, see for example [12]. We are currently exploring generalizations of our framework to other decision tasks such as distance-learning [16] and online convex programming [17].

## References

[1] H. H. Bauschke and J. M. Borwein. On projection algorithms for solving convex feasibility problems. *SIAM Review*, 1996.

[2] Y. Censor and S. A. Zenios. *Parallel Optimization.*. Oxford University Press, 1997.

[3] K. Crammer and Y. Singer. Ultraconservative online algorithms for multiclass problems. *Jornal of Machine Learning Research*, 3:951–991, 2003.

[4] Y. Freund and R. E. Schapire. Large margin classification using the perceptron algorithm. *Machine Learning*, 37(3):277–296, 1999.

[5] C. Gentile. A new approximate maximal margin classification algorithm. *Journal of Machine Learning Research*, 2:213–242, 2001.

[6] C. Gentile and M. Warmuth. Linear hinge loss and average margin. In NIPS'98.

[7] D. P. Helmbold, R. E. Schapire, Y. Singer, and M. K. Warmuth. A comparison of new and old algorithms for a mixture estimation problem. In COLT'95.

[8] M. Herbster. Learning additive models online with fast evaluating kernels. In COLT'01.

[9] J. Kivinen, D. P. Helmbold, and M. Warmuth. Relative loss bounds for single neurons. *IEEE Transactions on Neural Networks*, 10(6):1291–1304, 1999.

[10] J. Kivinen, A. J. Smola, and R. C. Williamson. Online learning with kernels. In NIPS'02.

[11] J. Kivinen and M. K. Warmuth. Exponentiated gradient versus gradient descent for linear predictors. *Information and Computation*, 132(1):1–64, January 1997.

[12] J. Kivinen and M. K. Warmuth. Relative loss bounds for multidimensional regression problems. *Journal of Machine Learning*, 45(3):301–329, July 2001.

[13] N. Klasner and H. U. Simon. From noise-free to noise-tolerant and from on-line to batch learning. In COLT'95.

[14] Y. Li and P. M. Long. The relaxed online maximum margin algorithm. *Machine Learning*, 46(1–3):361–387, 2002.

[15] V. N. Vapnik. *Statistical Learning Theory*. Wiley, 1998.

[16] E. Xing, A. Y. Ng, M. Jordan, and S. Russel. Distance metric learning, with application to clustering with side-information. In NIPS'03.

[17] M. Zinkevich. Online convex programming and generalized infinitesimal gradient ascent. In ICML'03.
